# Perceptron Learning of SAT

**Alex Flint**
Department of Engineering Science
University of Oxford
alexf@robots.ox.ac.uk

**Matthew B. Blaschko**
Center for Visual Computing
Ecole Centrale Paris
matthew.blaschko@inria.fr

## Abstract

Boolean satisfiability (SAT) as a canonical NP-complete decision problem is one of the most important problems in computer science. In practice, real-world SAT sentences are drawn from a distribution that may result in efficient algorithms for their solution. Such SAT instances are likely to have shared characteristics and substructures. This work approaches the exploration of a family of SAT solvers as a learning problem. In particular, we relate polynomial time solvability of a SAT subset to a notion of margin between sentences mapped by a feature function into a Hilbert space. Provided this mapping is based on polynomial time computable statistics of a sentence, we show that the existance of a margin between these data points implies the existance of a polynomial time solver for that SAT subset based on the Davis-Putnam-Logemann-Loveland algorithm. Furthermore, we show that a simple perceptron-style learning rule will find an optimal SAT solver with a bounded number of training updates. We derive a linear time computable set of features and show analytically that margins exist for important polynomial special cases of SAT. Empirical results show an order of magnitude improvement over a state-of-the-art SAT solver on a hardware verification task.

## 1 Introduction

SAT was originally shown to be a canonical NP-complete problem in Cook's seminal work [5]. SAT is of practical interest for solving a number of critical problems in applications such as theorem proving [8], model checking [2], planning [19], and bioinformatics [22]. That it is NP-complete indicates that an efficient learning procedure is unlikely to exist to solve arbitrary instances of SAT. Nevertheless, SAT instances resulting from real world applications are likely to have shared characteristics and substructures. We may view them as being drawn from a distribution over SAT instances, and for key problems this distribution may be benign in that a learning algorithm can enable quick determination of SAT. In this work, we explore the application of a perceptron inspired learning algorithm applied to branching heuristics in the Davis-Putnam-Logemann-Loveland algorithm [8, 7].

The Davis-Putnam-Logemann-Loveland (DPLL) algorithm formulates SAT as a search problem, resulting in a valuation of variables that satisfies the sentence, or a tree resolution refutation proof indicating that the sentence is not satisfiable. The branching rule in this depth-first search procedure is a key determinant of the efficiency of the algorithm, and numerous heuristics have been proposed in the SAT literature [15, 16, 26, 18, 13]. Inspired by the recent framing of learning as search optimization [6], we explore here the application of a perceptron inspired learning rule to application specific samples of the SAT problem. Efficient learning of SAT has profound implications for algorithm development across computer science as a vast number of important problems are polynomial time reducable to SAT.

A number of authors have considered learning branching rules for SAT solvers. Ruml applied reinforcement learning to find valuations of satisfiable sentences [25]. An approach that has performed well in SAT competitions in recent years is based on selecting a heuristic from a fixed set and apply-

ing it on a per-sentence basis [27, 17]. The relationship between learnability and NP-completeness has long been considered in the literature, e.g. [20]. Closely related to our approach is the learning as search optimization framework [6]. That approach makes perceptron-style updates to a heuristic function in $A^*$ search, but to our knowledge has not been applied to SAT, and requires a level of supervision that is not available in a typical SAT setting. A similar approach to learning heuristics for search was explored in [12].

## 2 Theorem Proving as a Search Problem

The SAT problem [5] is to determine whether a sentence $\Omega$ in propositional logic is satisfiable. First we introduce some notation. A binary variable $q$ takes on one of two possible values, $\{0, 1\}$. A literal $p$ is a proposition of the form $q$ (a "positive literal") or $\neg q$ (a "negative literal"). A clause $\omega_k$ is a disjunction of $n_k$ literals, $p_1 \vee p_2 \vee \cdots \vee p_{n_k}$. A unit clause contains exactly one literal. A sentence $\Omega$ in conjunctive normal form (CNF) [15] is a conjunction of $m$ clauses, $\omega_1 \wedge \omega_2 \wedge \cdots \wedge \omega_m$.

A *valuation* $B$ for $\Omega$ assigns to each variable in $\Omega$ a value $b_i \in \{0, 1\}$. A variable is *free* under $B$ if $B$ does not assign it a value. A sentence $\Omega$ is *satisfiable* iff there exists a valuation under which $\Omega$ is true. CNF is considered a canonical representation for automated reasoning systems. All sentences in propositional logic can be transformed to CNF [15].

### 2.1 The Davis–Putnam–Logemann–Loveland algorithm

Davis *et al.* [7] proposed a simple procedure for recognising satisfiabile CNF sentences on $N$ variables. Their algorithm is essentially a depth first search over all possible $2^N$ valuations over the input sentence, with specialized criteria to prune the search and transformation rules to simplify the sentence. We summarise the DPLL procedure below.

> **if** $\Omega$ contains only unit clauses and no contradictions **then**
>     **return** YES
> **end if**
> **if** $\Omega$ contains an empty clause **then**
>     **return** NO
> **end if**
> **for all** unit clauses $\omega \in \Omega$ **do**
>     $\Omega := \text{UnitPropagate}(\Omega, \omega)$
> **end for**
> **for all** literals $p$ such that $\neg p \notin \Omega$ **do**
>     remove all clauses containing $p$ from $\Omega$
> **end for**
> $p := \text{PickBranch}(\Omega)$
> **return** DPLL$(\Omega \wedge p) \vee$ DPLL$(\Omega \wedge \neg p)$

UnitPropagate simplifies $\Omega$ under the assumption $p$. PickBranch applies a heuristic to choose a literal in $\Omega$. Many modern SAT algorithms contain the DPLL procedure at their core [15, 16, 26, 18, 13], including top performers at recent SAT competitions [21]. Much recent work has focussed on choosing heuristics for the selection of branching literals since good heuristics have been empirically shown to reduce processing time by several orders of magnitude [28, 16, 13].

In this paper we learn heuristics by optimizing over a family of the form, $\operatorname{argmax}_p f(x, p)$ where $x$ is a node in the search tree, $p$ is a candidate literal, and $f$ is a priority function mapping possible branches to real numbers. The state $x$ will contain at least a CNF sentence and possibly pointers to ancestor nodes or statistics of the local search region. Given this relaxed notion of the search state, we are unaware of any branching heuristics in the literature that cannot be expressed in this form. We explicitly describe several in section 4.

## 3 Perceptron Learning of SAT

We propose to learn $f$ from a sequence of sentences drawn from some distribution determined by a given application. We identify $f$ with an element of a Hilbert space, $\mathcal{H}$, the properties of which

are determined by a set of statistics polynomial time computable from a SAT instance, $\Omega$. We apply stochastic updates to our estimate of $f$ in order to reduce our expected search time. We use $x_j$ to denote a node that is visited in the application of the DPLL algorithm, and $\phi_i(x_j)$ to denote the feature map associated with instantiating literal $p_i$. Using reproducing kernel Hilbert space notation, our decision function at $x_j$ takes the form

$$\operatorname*{argmax}_i \langle f, \phi_i(x_j) \rangle_{\mathcal{H}}. \tag{1}$$

We would like to learn $f$ such that the expected search time is reduced. We define $y_{ij}$ to be $+1$ if the instantiation of $p_i$ at $x_j$ leads to the shortest possible proof, and $-1$ otherwise. Our learning procedure therefore will ideally learn a setting of $f$ that only instantiates literals for which $y_{ij}$ is $+1$. We define a margin in a standard way:

$$\max \gamma \text{ s.t. } \langle f, \phi_i(x_j) \rangle_{\mathcal{H}} - \langle f, \phi_k(x_l) \rangle_{\mathcal{H}} \geq \gamma \quad \forall \{(i,j)|y_{ij} = +1\}, \{(k,l)|y_{kl} = -1\} \tag{2}$$

### 3.1 Restriction to Satisfiable Sentences

If we had access to all $y_{ij}$, the application of any binary learning algorithm to the problem of learning SAT would be straightforward. Unfortunately, the identity of $y_{ij}$ is only known in the worst case after an exhaustive enumeration of all $2^N$ variable assignments. We do note, however, that the DPLL algorithm is a depth–first search over literal valuations. Furthermore, for satisfiable sentences the length of the shortest proof is bounded by the number of variables. Consequently, in this case, all nodes visited on a branch of the search tree that resolved to unsatisfiable have $y_{ij} = -1$ and the nodes on the branch leading to satisfiable have $y_{ij} = +1$. We may run the DPLL algorithm with a current setting of $f$ and if the sentence is satisfiable, update $f$ using the inferred $y_{ij}$.

This learning framework is capable of computing in polynomial time valuations of satisfiable sentences in the following sense.

**Theorem 1** $\exists$ *a polynomial time computable $\phi$ with $\gamma > 0$ $\iff$ $\Omega$ belongs to a subset of satisfiable sentences for which there exists a polynomial time algorithm to find a valid valuation.*

**Proof** Necessity is shown by noting that the $\operatorname{argmax}$ in each step of the DPLL algorithm is computable in time polynomial in the sentence length by computing $\phi$ for all literals, and that there exists a setting of $f$ such that there will be at most a number of steps equal to the number of variables.

Sufficiency is shown by noting that we may run the polynomial algorithm to find a valid valuation and use that valuation to construct a feature space with $\gamma \geq 0$ in polynomial time. Concretely, choose a canonical ordering of literals indexed by $i$ and let $\phi_i(x_j)$ be a scalar. Set $\phi_i(x_j) = +i$ if literal $p_i$ is instantiated in the solution found by the polynomial algorithm, $-1$ otherwise. When $f = 1$, $\gamma = 2$. $\square$

**Corollary 1** $\exists$ *polynomial time computable feature space with $\gamma > 0$ for SAT $\iff$ $P = NP$*

**Proof** If $P = NP$ there is a polynomial time solution to SAT, meaning that there is a polynomial time solution to finding valuations satisfiable sentences. For satisfiable sentences, this indicates that there is a non-negative margin. For unsatisfiable sentences, either a proof exists with length less than the number of variables, or we may terminate the DPLL procedure after $N + 1$ steps and return unsatisfiable. $\square$

While Theorem 1 is positive for finding variable settings that satisfy sentences, unsatisfiable sentences remain problematic when we are unsure that there exists $\gamma > 0$ or if we have an incorrect setting of $f$. We are unaware of an efficient method to determine all $y_{ij}$ for visited nodes in proofs of unsatisfiable sentences. However, we expect that similar substructures will exist in satisfiable and unsatisfiable sentences resulting from the same application. Early iterations of our learning algorithm will mistakenly explore branches of the search tree for satisfiable sentences and these branches will share important characteristics with inefficient branches of proofs of unsatisfiability. Consequently, proofs of unsatisfiability may additionally benefit from a learning procedure applied *only* to satisfiable sentences. In the case that we analyitically know that $\gamma > 0$ and we have a correct setting of $f$, we may use the termination procedure in Corollary 1.

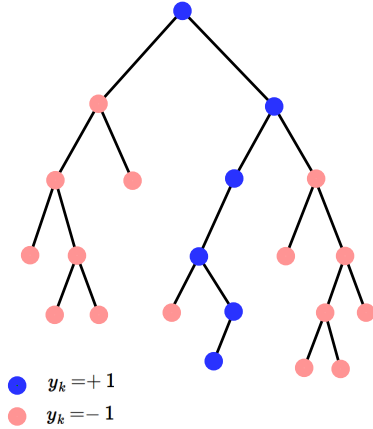

$y_k = +1$
$y_k = -1$

Figure 1: Generation of training samples from the search tree. Nodes labeled in red result in backtracking and therefore have negative label, while those coloured blue lie on the path to a proof of satisfiability.

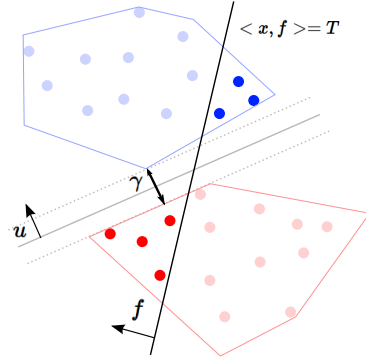

$< x, f >= T$

Figure 2: Geometry of the feature space. Positive and negative nodes are separated by a margin of $\gamma$. Given the current estimate of $f$, a threshold, $T$, is selected as described in section 3.2. The positive nodes with a score less than $T$ are averaged, as are negative nodes with a score greater than $T$. The resulting means lie within the respective convex hulls of the positive and negative sets, ensuring that the geometric conditions of the proof of Theorem 2 are fulfilled.

### 3.2 Davis-Putnam-Logemann-Loveland Stochastic Gradient

We use a modified perceptron style update based on the learning as search optimization framework proposed in [6]. In contrast to that work, we do not have a notion of "good" and "bad" nodes at each search step. Instead, we must run the DPLL algorithm to completion with a fixed model, $f_t$. We know that nodes on a path to a valuation that satisfies the sentence have positive labels, and those nodes that require backtracking have negative labels (Figure 1). If the sentence is satisfiable, we may compute a DPLL stochastic gradient, $\nabla_{\text{DPLL}}$, and update $f$. We define two sets of nodes, $\mathcal{S}_+$ and $\mathcal{S}_-$, such that all nodes in $\mathcal{S}_+$ have positive label and lower score than all nodes in $\mathcal{S}_-$ (Figure 2). In this work, we have used the sufficient condition of defining these sets by setting a score threshold, $T$, such that $f_k(\phi_i(x_j)) < T \ \forall (i,j) \in \mathcal{S}_+$, $f_k(\phi_i(x_j)) > T \ \forall (i,j) \in \mathcal{S}_-$ , and $|\mathcal{S}_+| \times |\mathcal{S}_-|$ is maximized. The DPLL stochastic gradient update is defined as follows:

$$\nabla_{\text{DPLL}} = \sum_{(i,j) \in \mathcal{S}_-} \frac{\phi_i(x_j)}{|\mathcal{S}_-|} - \sum_{(k,l) \in \mathcal{S}_+} \frac{\phi_k(x_l)}{|\mathcal{S}_+|}, \qquad f_{t+1} = f_t - \eta \nabla_{\text{DPLL}} \qquad (3)$$

where $\eta$ is a learning rate. While poor settings of $f_0$ may result in a very long proof before learning can occur, we show in Section 4 that we can initialize $f_0$ to emulate the behavior of current state-of-the-art SAT solvers. Subsequent updates improve performance over the baseline.

We define $R$ to be a positive real value such that $\forall i, j, k, l \ \ \|\phi_i(x_j) - \phi_k(x_l)\| \leq R$

**Theorem 2** *For any training sequence that is separable by a margin of size $\gamma$ with $\|f\| = 1$, using the update rule in Equation* (3) *with $\eta = 1$, the number of errors (updates) made during training on satisfiable sentences is bounded above by $R^2/\gamma^2$.*

**Proof** Let $f_1(\phi(x)) = 0 \ \forall \phi(x)$. Considering the $k$th update,

$$\|f_{k+1}\|^2 = \|f_k - \nabla_{\text{DPLL}}\|^2 = \|f_k\|^2 - 2\langle f_k, \nabla_{\text{DPLL}}\rangle + \|\nabla_{\text{DPLL}}\|^2 \leq \|f_k\|^2 + 0 + R^2. \quad (4)$$

We note that it is the case that $\langle f_k, \nabla_{\text{DPLL}}\rangle \geq 0$ for any selection of training examples such that the average of the negative examples score higher than the average of the positive examples generated by running a DPLL search. It is possible that some negative examples with lower scores than the some positive nodes will be visited during the depth first search of the DPLL algorithm, but we are guaranteed that at least one of them will have higher score. Similarly, some positive examples may have higher scores than the highest scoring negative example. In both cases, we may simply discard

| Feature | Dimensions | Description |
|---|---|---|
| is-positive | 1 | 1 if $p$ is positive, 0 otherwise |
| lit-unit-clauses | 1 | $C_1(p)$, occurences of literal in unit clauses |
| var-unit-clauses | 1 | $C_1(q)$, occurences of variable in unit clauses |
| lit-counts | 3 | $C_i(p)$ for $i = 2, 3, 4$, occurences in small clauses |
| var-counts | 3 | $C_i(q)$ for $i = 2, 3, 4$, as above, by variable |
| bohm-max | 3 | $\max(C_i(p), C_i(\neg p)), i = 2, 3, 4$ |
| bohm-min | 3 | $\max(C_i(p), C_i(\neg p)), i = 2, 3, 4$ |
| lit-total | 1 | $C(p)$, total occurences by literal |
| neg-lit-total | 1 | $C(\neg p)$, total occurences of negated literal |
| var-total | 1 | $C(q)$, total occurences by variable |
| lit-smallest | 1 | $C_m(p)$, where $m$ is the size of the smallest unsatisfied clause |
| neg-lit-smallest | 1 | $C_m(\neg p)$, as above, for negated literal |
| jw | 1 | $J(p)$ Jeroslow–Wang cue, see main text |
| jw-neg | 1 | $J(\neg p)$ Jeroslow–Wang cue, see main text |
| activity | 1 | minisat activity measure |
| time-since-active | 1 | $t - T(p)$ time since last activity (see main text) |
| has-been-active | 1 | 1 if this $p$ has ever appeared in a conflict clause; 0 otherwise |

Figure 3: Summary of our feature space. Features are computed as a function of a sentence $\Omega$ and a literal $p$. $q$ implicitly refers to the variable within $p$.

such instances from the training algorithm (as described in Section 3.2) guaranteeing the desired inequality. By induction, $\|f_{k+1}\|^2 \leq kR^2$.

Let $u$ be an element of $\mathcal{H}$ that obtains a margin of $\gamma$ on the training set. We next obtain a lower bound on $\langle u, f_{k+1} \rangle = \langle u, f_k \rangle - \langle u, \nabla_{\text{DPLL}} \rangle \geq \langle u, f_k \rangle + \gamma$. That $-\langle u, \nabla_{\text{DPLL}} \rangle \geq \gamma$ follows from the fact that the means of the positive and negative training examples lie in the convex hull of the positive and negative sets, respectively, and that $u$ achieves a margin of $\gamma$. By induction, $\langle u, f_{k+1} \rangle \geq k\gamma$.

Putting the two results together gives $\sqrt{k}R \geq \|f_{k+1}\| \geq \langle u, f_{k+1} \rangle \geq k\gamma$ which, after some algebra, yields $k \leq (R/\gamma)^2$. □

The proof of this theorem closely mirrors those of the mistake bounds in [24, 6]. We note also that an extension to approximate large-margin updates is straightforward to implement, resulting in an alternate mistake bound (c.f. [6, Theorem 4]). For simplicity we consider only the perceptron style updates of Equation (3) in the sequel.

# 4 Feature Space

In this section we describe our feature space. Recall that each node $x_j$ consists of a CNF sentence $\Omega$ together with a valuation for zero or more variables. Our feature function $\phi(x, p)$ maps a node $x$ and a candidate branching literal $p$ to a real vector $\phi$. Many heuristics involve counting occurences of literals and variables. For notational convenience let $C(p)$ be the number of occurences of $p$ in $\Omega$ and let $C_k(p)$ be the number of occurrences of $p$ among clauses of size $k$. Table 4 summarizes our feature space.

## 4.1 Relationship to previous branching heuristics

Many branching heuristics have been proposed in the SAT literature [28, 13, 18, 26]. Our features were selected from the most successful of these and our system is hence able to emulate many other systems for particular priority functions $f$.

**Literal counting.** Silva [26] suggested two simple heuristics based directly on literal counts. The first was to always branch on the literal that maximizes $C(p)$ and the second was to maximize $C(p) + C(\neg p)$. Our features "lit-total" and "neg-lit-total" capture these cues.
**MOM**. Freeman [13] proposed a heuristic that identified the size of the smallest unsatisfied clause, $m = \min |\omega|, \omega \in \Omega$, and then identified the literal appearing most frequently amongst clauses of size $m$. This is the motivation for our features "lit-smallest" and "neg-lit-smallest".
**BOHM**. Bohm [3] proposed a heuristic that selects the literal maximizing

$$\alpha \max\Big(C_k(p, x_j), C_k(\neg p, x_j)\Big) + \beta \min\Big(C_k(p, x_j), C_k(\neg p, x_j)\Big), \tag{5}$$

with $k = 2$, or in the case of a tie, with $k = 3$ (and so on until all ties are broken). In practice we found that ties are almost always broken by considering just $k \le 4$; hence we include "bohm-max" and "bohm-min" in our feature space.

**Jeroslow–Wang**. Jerosolow and Wang [18] proposed a voting scheme in which clauses vote for their components with weight $2^{-k}$, where $k$ is the length of the clause. The total votes for a literal $p$ is

$$J(p) = \sum 2^{-|\omega|} \tag{6}$$

where the sum is over clauses $\omega$ that contain $p$. The Jeroslow–Wang rule chooses branches that maximize $J(p)$. Three variants were studied by Hooker [16]. Our features "jw" and "jw-neg" are sufficient to span the original rule as well as the variants.

**Dynamic activity measures.** Many modern SAT solvers use boolean constraint propagation (BCP) to speed up the search process [23]. One component of BCP generates new clauses as a result of conflicts encountered during the search. Several modern SAT solvers use the time since a variable was last added to a conflict clause to measure the "activity" of that variable . Empirically, resolving variables that have most recently appeared in conflict clauses results in an efficient search[14]. We include several activity–related cues in our feature vector, which we compute as follows. Each decision is is given a sequential time index $t$. After each decision we update the most–recent– activity table $T(p) := t$ for each $p$ added to a conflict clause during that iteration. We include the difference between the current iteration and the last iteration at which a variable was active in the feature "time-since-active". We also include the boolean feature "has-been-active" to indicate whether a variable has ever been active. The feature "activity" is a related cue used by minisat [10].

## 5 Polynomial special cases

In this section we discuss special cases of SAT for which polynomial–time algorithms are known. For each we show that a margin exists in our feature space.

### 5.1 Horn

A Horn clause [4] is a disjunction containing at most one positive literal, $\neg q_1 \vee \cdots \vee \neg q_{k-1} \vee q_k$. A sentence $\Omega$ is a Horn formula iff it is a conjunction of Horn clauses. There are polynomial time algorithms for deciding satisfiability of Horn formulae [4, 9]. One simple algorithm based on unit propagation [4] operates as follows. If there are no unit clauses in $\Omega$ then $\Omega$ is trivially satisfiable by setting all variables to false. Otherwise, let $\{p\}$ be a unit clause in $\Omega$. Delete any clause from $\Omega$ that contains $p$ and remove $\neg p$ wherever it appears. Repeat until either a trivial contradiction $q \wedge \neg q$ is produced (in which case $\Omega$ is unsatisfiable) or until no further simplification is possible (in which case $\Omega$ is satisfiable) [4].

**Theorem 3** *There is a margin for Horn clauses in our feature space.*

**Proof** We will show that there is a margin for Horn clauses in our feature space by showing that for a particular priority function $f_0$, our algorithm will emulate the unit propagation algorithm above. Let $f_0$ be zero everywhere except for the following elements:[1]"is-positive" $= -\epsilon$, "lit-unit-clauses" $= 1$. Let $H$ be the decision heuristic corresponding to $f_0$. Consider a node $x$ and let $\Omega$ be the input sentence $\Omega_0$ simplified according to the (perhaps partial) valuation at $x$. If $\Omega$ contains no unit clauses then clearly $\langle \phi(x, p), f_0 \rangle$ will be maximized for a negative literal $p = \neg q$. If $\Omega$ does contain unit clauses then for literals $p$ which appear in unit clauses we have $\langle \phi(x, p), f_0 \rangle \ge 1$, while for all other literals we have $\langle \phi(x, p), f_0 \rangle < 1$. Therefore $H$ will select a unit literal if $\Omega$ contains one.

For satisfiable $\Omega$, this exactly emulates the unit propagation algorithm, and since that algorithm never back–tracks [4], our algorithm makes no mistakes. For unsatisfiable $\Omega$ our algorithm will behave as follows. First note that every sentence encountered contains at least one unit clause, since, if not, that sentence would be trivially satisfiable by setting all variables to false and this would contradict the assumption that $\Omega$ is unsatisfiable. So at each node $x$, the algorithm will first branch on some unit clause $p$, then later will back–track to $x$ and branch on $\neg p$. But since $p$ appears in a unit clause at $x$ this will immediately generate a contradiction and no further nodes will be expanded along that path. Therefore the algorithm expands no more than $2N$ nodes, where $N$ is the length of $\Omega$. $\quad\square$

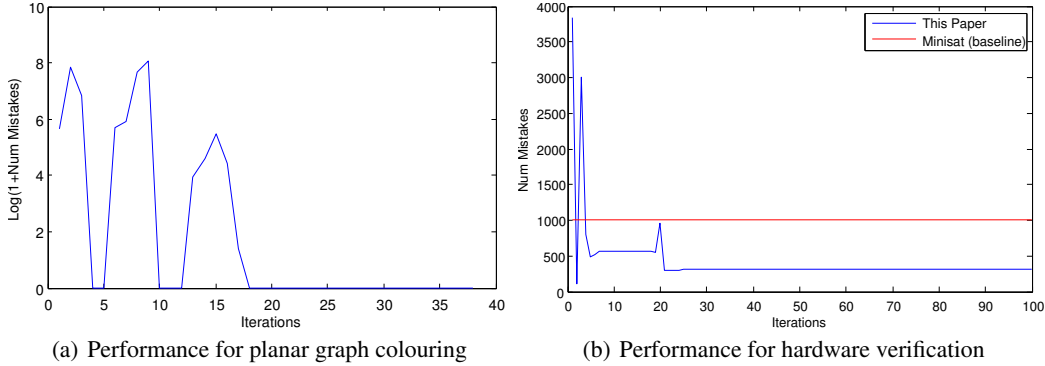

(a) Performance for planar graph colouring     (b) Performance for hardware verification

Figure 4: Results for our algorithm applied to (a) planar graph colouring; (b) hardware verification. Both figures show the mistake rate as a function of the training iteration. In figure (a) we report the mistake rate on the current training example since no training example is ever repeated, while in figure (b) it is computed on a seperate validation set (see figure 5). The red line shows the performance of minisat on the validation set (which does not change over time).

## 5.2   2–CNF

A 2–CNF sentence is a CNF sentence in which every clause contains exactly two literals. In this section we show that a function exists in our feature space for recognising satisfiable 2–CNF sentences in polynomial time.

A simple polynomial–time solution to 2–CNF proposed by Even *et al.* [11] operates as follows. If there are no unit clauses in $\Omega$ then pick any literal and add it to $\Omega$. Otherwise, let $\{p\}$ be a unit clause in $\Omega$ and apply unit propagation to $p$ as described in the previous section. If a contradiction is generated then back–track to the last branch and negate the literal added there. If there is no such branch, then $\Omega$ is unsatisfiable. Even *et al.* showed that this algorithm never back–tracks over more than one branch, and therefore completes in polynomial time.

**Theorem 4** *Under our feature space, $\mathcal{H}$ contains a priority function that recognizes 2–SAT sentences in polynomial time.*

**Proof**  By construction. Let $f_0$ be a weight vector with all elements set to zero except for the element corrersponding to the "appears-in-unit-clause" feature, which is set to 1. When using this weight vector, our algorithm will branch on a unit literal whenever one is present. This exactly emulates the behaviour of the algorithm due to Even *et al.* described above, and hence completes in polynomial time for all 2–SAT sentences.     □

## 6   Empirical Results

**Planar Graph Colouring:** We applied our algorithm on the problem of planar graph colouring, for which polynomial time algorithms are known [1]. Working in this domain allowed us to generate an unlimited number of problems with a consistent but non–trivial structure on which to validate our algorithm. By allowing up to four colours we also ensured that all instances were satisfiable [1].

We generated instances as follows. Starting with an empty $L \times L$ grid we sampled $K$ cells at random and labelled them $1 \ldots K$. We then repeatedly picked a labelled cell with at least one unlabelled neighbour and copied its label to its neighbour until all cells were labelled. Next we formed a $K \times K$ adjacency matrix $A$ with $A_{ij} = 1$ iff there is a pair of adjacent cells with labels $i$ and $j$. Finally we generated a SAT sentence over $4K$ variables (each variable corresponds to a particular colouring of a particular vertex), with clauses expressing the constraints that each vertex must be assigned one and only one colours and that no pair of adjacent vertices may be assigned the same colour.

In our experiments we used $K = 8, L = 5$ and a learning rate of $0.1$. We ran 40 training iterations of our algorithm. No training instance was repeated. The number of mistakes (branching decision that

| Training | | Validation | |
|---|---|---|---|
| Problem | Clauses | Problem | Clauses |
| ferry11 | 26106 | ferry10 | 20792 |
| ferry11u | 25500 | ferry10u | 20260 |
| ferry9 | 16210 | ferry8 | 12312 |
| ferry9u | 15748 | ferry8u | 11916 |
| ferry12u | 31516 | ferry12 | 32200 |

Figure 5: Instances in training and validation sets.

were later reversed by back–tracking) made at each iteration is shown in figure 4(a). Our algorithm converged after 18 iterations and never made a mistake after that point.

**Hardware Verification:** We applied our algorithm to a selection of problems from a well–known SAT competition [21]. We selected training and validation examples from the same suite of problems; this is in line with our goal of learning the statistical structure of particular subsets of SAT problems. The problems selected for training and validation are from the 2003 SAT competition and are listed in figure 5.

Due to the large size of these problems we extended an existing high–performance SAT solver, minisat [10], replacing its decision heuristic with our perceptron strategy. We executed our algorithm on each training problem sequentially for a total of 8 passes through the training set (40 iterations in total). We performed a perceptron update (3) after solving each problem. After each update we evaluated the current priority function on the entire validation set. The average mistake rate on the validation set are shown for each training iteration in figure 4(b).

# 7 Discussion

Section 6 empirically shows that several important theoretical results of our learning algorithm hold in practice. The experiments reported in Figure 4(a) show in practice that for a polynomial time solvable subset of SAT, the algorithm indeed has a bounded number of mistakes during training. Planar graph colouring is a known polynomial time computable problem, but it is difficult to characterize theoretically and an automated theorem prover was employed in the proof of polynomial solvability. The hardware verification problem explored in Figure 4(b) shows that the algorithm learns a setting of $f$ that gives performance an order of magnitude faster than the state of the art Minisat solver. It does so after relatively few training iterations and then maintains good performance.

Several approaches present themselves as good opportunites of extensions to learning SAT. In this work, we argued that learning on positive examples is sufficient if the subset of SAT sentences generated by our application has a positive margin. However, it is of interest to consider learning in the absense of a positive margin, and learning may be accelerated by making updates based on unsatisfiable sentences. One potential approach would be to consider a stochastic finite difference approximation to the risk gradient by running the DPLL algorithm a second time with a perturbed $f$. Additionally, we may consider updates to $f$ during a run of the DPLL algorithm when the algorithm backtracks from a branch of the search tree for which we can prove that all $y_{ij} = -1$. This, however, requires care in ensuring that the implicit empirical risk minimization is not biased.

In this work, we have shown that a perceptron-style algorithm is capable of learning all polynomial solvable SAT subsets in bounded time. This has important implications for learning real-world SAT applications such as theorem proving, model checking, planning, hardware verification, and bioinformatics. We have shown empirically that our theoretical results hold, and that state-of-the-art computation time can be achieved with our learning rule on a real-world hardware verification problem. As SAT is a canonical NP-complete problem, we expect that the efficient solution of important subsets of SAT may have much broader implications for the solution of many real-world problems.

**Acknowledgements:** This work is partially funded by the European Research Council under the European Community's Seventh Framework Programme (FP7/2007-2013)/ERC Grant agreement number 259112, and by the Royal Academy of Engineering under the Newton Fellowship Alumni Scheme.

## Footnotes

[1] For concreteness let $\epsilon = \frac{1}{K+1}$ where $K$ is the length of the input sentence $\Omega$

# References

[1] K. Appel, W. Haken, and J. Koch. Every planar map is four colorable. *Illinois J. Math*, 21(3):491 – 567, 1977.

[2] A. Biere, A. Cimatti, E. M. Clarke, and Y. Zhu. Symbolic model checking without BDDs. In *International Conference on Tools and Algorithms for Construction and Analysis of Systems*, pages 193–207, 1999.

[3] M. Buro and H. K. Buning. Report on a sat competition. 1992.

[4] C.-L. Chang and R. C.-T. Lee. *Symbolic Logic and Mechanical Theorem Proving*. Academic Press, Inc., Orlando, FL, USA, 1st edition, 1997.

[5] S. A. Cook. The complexity of theorem-proving procedures. In *Proceedings of the third annual ACM symposium on Theory of computing*, STOC '71, pages 151–158, New York, NY, USA, 1971. ACM.

[6] H. Daumé, III and D. Marcu. Learning as search optimization: approximate large margin methods for structured prediction. In *International Conference on Machine learning*, pages 169–176, 2005.

[7] M. Davis, G. Logemann, and D. Loveland. A machine program for theorem-proving. *Commun. ACM*, 5:394–397, July 1962.

[8] M. Davis and H. Putnam. A computing procedure for quantification theory. *J. ACM*, 7:201–215, 1960.

[9] W. F. Dowling and J. H. Gallier. Linear-time algorithms for testing the satisfiability of propositional horn formulae. *The Journal of Logic Programming*, 1(3):267 – 284, 1984.

[10] N. Eén and N. Sörensson. An extensible sat-solver. In *Theory and Applications of Satisfiability Testing*, pages 333–336. 2004.

[11] S. Even, A. Itai, and A. Shamir. On the complexity of time table and multi-commodity flow problems. In *Symposium on Foundations of Computer Science*, pages 184–193, 1975.

[12] M. Fink. Online learning of search heuristics. *Journal of Machine Learning Research - Proceedings Track*, 2:114–122, 2007.

[13] J. W. Freeman. *Improvements to propositional satisfiability search algorithms*. PhD thesis, University of Pennsylvania, 1995.

[14] E. Goldberg and Y. Novikov. Berkmin: A fast and robust sat-solver. In *Design, Automation and Test in Europe Conference and Exhibition, 2002. Proceedings*, pages 142 –149, 2002.

[15] J. Harrison. *Handbook of Practical Logic and Automated Reasoning*. Cambridge University Press, 2009.

[16] J. N. Hooker and V. Vinay. Branching rules for satisfiability. *Journal of Automated Reasoning*, 15:359–383, 1995.

[17] F. Hutter, D. Babic, H. H. Hoos, and A. J. Hu. Boosting verification by automatic tuning of decision procedures. In *Proceedings of the Formal Methods in Computer Aided Design*, pages 27–34, 2007.

[18] R. G. Jeroslow and J. Wang. Solving propositional satisfiability problems. *Annals of Mathematics and Artificial Intelligence*, 1:167–187, 1990.

[19] H. A. Kautz. Deconstructing planning as satisfiability. In *Proceedings of the Twenty-first National Conference on Artificial Intelligence (AAAI-06)*, 2006.

[20] M. Kearns, M. Li, L. Pitt, and L. Valiant. On the learnability of boolean formulae. In *Proceedings of the nineteenth annual ACM symposium on Theory of computing*, pages 285–295, 1987.

[21] D. Le Berra and O. Roussel. Sat competition 2009. http://www.satcompetition.org/2009/.

[22] I. Lynce and J. a. Marques-Silva. Efficient haplotype inference with boolean satisfiability. In *Proceedings of the 21st national conference on Artificial intelligence - Volume 1*, pages 104–109. AAAI Press, 2006.

[23] M. Moskewicz, C. Madigan, Y. Zhao, L. Zhang, and S. Malik. Chaff: engineering an efficient sat solver. In *Design Automation Conference, 2001. Proceedings*, pages 530 – 535, 2001.

[24] F. Rosenblatt. The Perceptron: A probabilistic model for information storage and organization in the brain. *Psychological Review*, 65:386–408, 1958.

[25] W. Ruml. *Adaptive Tree Search*. PhD thesis, Harvard University, 2002.

[26] J. a. P. M. Silva. The impact of branching heuristics in propositional satisfiability algorithms. In *Proceedings of the 9th Portuguese Conference on Artificial Intelligence: Progress in Artificial Intelligence*, EPIA '99, pages 62–74, London, UK, 1999. Springer-Verlag.

[27] L. Xu, F. Hutter, H. H. Hoos, and K. Leyton-Brown. Satzilla: portfolio-based algorithm selection for sat. *J. Artif. Int. Res.*, 32:565–606, June 2008.

[28] R. Zabih. A rearrangement search strategy for determining propositional satisfiability. In *in Proceedings of the National Conference on Artificial Intelligence*, pages 155–160, 1988.

